# Unsupervised learning models of primary cortical receptive fields and receptive field plasticity

**Andrew Saxe, Maneesh Bhand, Ritvik Mudur, Bipin Suresh, Andrew Y. Ng**
Department of Computer Science
Stanford University
{asaxe, mbhand, rmudur, bipins, ang}@cs.stanford.edu

## Abstract

The efficient coding hypothesis holds that neural receptive fields are adapted to the statistics of the environment, but is agnostic to the timescale of this adaptation, which occurs on both evolutionary and developmental timescales. In this work we focus on that component of adaptation which occurs during an organism's lifetime, and show that a number of unsupervised feature learning algorithms can account for features of normal receptive field properties across multiple primary sensory cortices. Furthermore, we show that the same algorithms account for altered receptive field properties in response to experimentally altered environmental statistics. Based on these modeling results we propose these models as phenomenological models of receptive field plasticity during an organism's lifetime. Finally, due to the success of the same models in multiple sensory areas, we suggest that these algorithms may provide a constructive realization of the theory, first proposed by Mountcastle [1], that a qualitatively similar learning algorithm acts throughout primary sensory cortices.

## 1 Introduction

Over the last twenty years, researchers have used a number of unsupervised learning algorithms to model a range of neural phenomena in early sensory processing. These models have succeeded in replicating many features of simple cell receptive fields in primary visual cortex [2, 3], as well as cochlear nerve fiber responses in the subcortical auditory system [4]. Though these algorithms do not perfectly match the experimental data (see [5]), they continue to improve in recent work (e.g. [6, 7]). However, each phenomenon has generally been fit by a different algorithm, and there has been little comparison of an individual algorithm's breadth in simultaneously capturing different types of data. In this paper we test whether a single learning algorithm can provide a reasonable fit to data from three different primary sensory cortices. Further, we ask whether such algorithms can account not only for typical data from normal environments but also for experimental data from animals raised with drastically different environmental statistics.

Our motivation for exploring the breadth of each learning algorithm's applicability is partly biological. Recent reviews of the experimental literature regarding the functional consequences of plasticity have remarked on the surprising similarity in plasticity outcomes across sensory cortices [8, 9]. These empirical results raise the possibility that a single phenomenological model of plasticity (a "learning algorithm" in our terminology) might account for receptive field properties independent of modality. Finding such a model, if it exists, could yield broad insight into early sensory processing strategies. As an initial step in this direction, we evaluate the match between current unsupervised learning algorithms and receptive field properties in visual, auditory, and somatosensory cortex. We find that many current algorithms achieve qualitatively similar matches to receptive field properties in all three modalities, though differences between the models and experimental data remain.

In the second part of this paper, we examine the sensitivity of these algorithms to changes in their input statistics. Most previous work that uses unsupervised learning algorithms to explain neural

receptive fields makes no claim about the relative contributions of adaptation on evolutionary as compared to developmental timescales, but rather models the end point of these complex processes, that is, the receptive field ultimately measured in the adult animal. In this work, we consider the alternative view that significant adaptation occurs during an organism's lifetime, i.e., that the learning algorithm operates predominantly during development rather than over the course of evolution.

One implication of lifetime adaptation is that experimental manipulations of early sensory experience should result in altered receptive field properties. We therefore ask whether current unsupervised learning algorithms can reproduce appropriately altered receptive field properties in response to experimentally altered inputs. Our results show that the same unsupervised learning algorithm can model normal and altered receptive fields, yielding an account of sensory receptive fields focused heavily on activity dependent plasticity processes operating during an organism's lifetime.

## 2    Modeling approach

We use the same three stage processing pipeline to model each modality; the first stage models peripheral end-receptors, namely rods and cones in the retina, hair cells in the cochlea, and mechanoreceptors in glabrous skin; the second stage crudely models subcortical processing as a whitening transformation of the data; and the third stage models cortical receptive field plasticity mechanisms as an unsupervised learning algorithm. We note that the first two stages cannot do justice to the complexities of subcortical processing, and the simple approximation built into these stages limits the quality of fit we can expect from the models.

We consider five unsupervised learning algorithms: independent component analysis [10], sparse autoencoder neural networks [11], restricted Boltzmann machines (RBMs) [12], K-means [13], and sparse coding [2]. These algorithms were chosen on two criteria. First, all of the algorithms share the property of learning a sparse representation of the input, though they clearly differ in their details, and have at least qualitatively been shown to yield Gabor-like filters when applied to naturalistic visual input. Second, we selected algorithms to span a number of reasonable approaches and popular formalisms, i.e., efficient coding ideas, backpropagation in artificial neural networks, probabilistic generative models, and clustering methods. As we will show in the rest of the paper, in fact these five algorithms turn out to yield very similar results, with no single algorithm being decisively better.

Each algorithm contains a number of parameters which control the learning process, which we fit to the experimental data by performing extensive grid searches through the parameter space. To obtain an estimate of the variability in our results, we trained multiple models at each parameter setting but with different randomly drawn datasets and different initial weights. All error bars are the standard error of the mean. The results reported in this paper are for the best-fitting parameter settings for each algorithm per modality. We worried that we might overfit the experimental data due to the large number of models we trained ($\approx 60,000$). As one check against this, we performed a cross-validation-like experiment by choosing the parameters of each algorithm to maximize the fit to one modality, and then evaluating the performance of these parameters on the other two modalities. We found that, though quantitatively the results are slightly worse as expected, qualitatively the results follow the same patterns of which phenomena are well-fit (see supplementary material). Because we have fit model parameters to experimental data, we cannot assess the efficiency of the resulting code. Rather, our aim is to evaluate the single learning algorithm hypothesis, which is orthogonal to the efficient coding hypothesis. A learning algorithm could potentially learn a non-efficient code, for instance, but nonetheless describe the establishment of receptive fields seen in adult animals. Details of the algorithms, parameters, and fitting methods can be found in the supplementary information. Results from our grid searches are available at `http://www.stanford.edu/~asaxe/rf_plasticity.html`.

## 3    Naturalistic experience and normal receptive field properties

In this section we focus on whether first-order, linear properties of neural responses can be captured by current unsupervised learning algorithms applied to naturalistic visual, auditory, and somatosensory inputs. Such a linear description of neural responses has been broadly studied in all sensory cortices [14, 15, 16, 17]. Though a more complete model would incorporate nonlinear components, these more sophisticated nonlinear models often have as their first step a convolution with a linear kernel (see [18] for an overview); and it is this kernel which we suggest might be learned over the course of development, by a qualitatively similar learning algorithm across modalities.

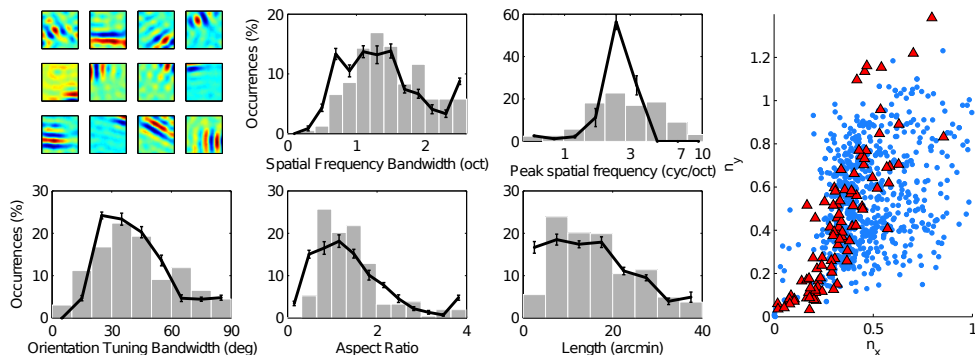

Figure 1: Top left: K-means bases learned from natural images. Histograms: Black lines show population statistics for K-means bases, gray bars show V1 simple cell data from Macaque. Far right: Distribution of receptive field shapes; Red triangles are V1 simple cells from [5], blue circles are K-means bases.

## 3.1 Primary visual cortex

A number of studies have shown that response properties in V1 can be successfully modeled using a variety of unsupervised learning algorithms [2, 19, 3, 12, 10, 6, 7]. We replicate these findings for the particular algorithms we employ and make the first detailed comparisons to experiment for the sparse autoencoder, sparse RBM, and K-means algorithms.

Our natural image dataset consists of ten gray scale images of outdoor scenes [2]. Multiple non-overlapping patches were sampled to form the first stage of our model, meant to approximate the response of rods and cones. This raw data was then whitened using PCA whitening in the second stage of the model, corresponding to retinal ganglion or LGN responses.[1] These inputs were supplied to each of the five learning algorithms.

Fig. 1 shows example bases learned by the K-means algorithm. All five algorithms learn localized, band-pass receptive field structures for a broad range of parameter settings, in qualitative agreement with the spatial receptive fields of simple cells in primary visual cortex. To better quantify the match, we compare five properties of model neuron receptive fields to data from macaque V1, namely the spatial frequency bandwidth, orientation tuning bandwidth, length, aspect ratio, and peak spatial frequency of the receptive fields. We compare population histograms of these metrics to those measured in macaque V1 by [14, 15] as reported in [3]. Fig. 1 shows these histograms for the best-fitting K-means bases according to the average L1 distance between model and data histograms. For all five algorithms, the histograms show general agreement with the distribution of parameters in primary visual cortex except for the peak spatial frequency, consistent with the results of previous studies for ICA and sparse coding [2, 3]. Additional plots for the other algorithms can be found in the supplementary materials.

Next, we compare the shape of simulated receptive fields to experimentally-derived receptive fields. As had been done for the experimental data, we fit Gabor functions to our simulated receptive fields and calculated the "normalized" receptive field sizes $n_x = \sigma_x f$ and $n_y = \sigma_y f$ where $\sigma_x$ is the standard deviation of the gaussian envelope along the axis with sinusoidal modulation, $\sigma_y$ is the stardard deviation of the gaussian envelope along the axis in which the filter is low pass, and $f$ is the frequency of the sinusoid. The parameters $n_x$ and $n_y$ measure the number of sinusoidal cycles that fit within an interval of length $\sigma_x$ and $\sigma_y$ respectively. Hence they capture the number of excitatory and inhibitory lobes of significant power in each receptive field. The right panel of Fig. 1 shows the distribution of $n_x$ and $n_y$ for K-means compared to those reported experimentally [5]. The model bases lie within the experimentally derived values, though our models fail to exhibit as much variability in shape as the experimentally-derived data. As had been noted for ICA and sparse coding in [5], all five of our algorithms fail to capture low frequency bases near the origin. These low frequency bases correspond to "blobs" with just a single excitatory region.

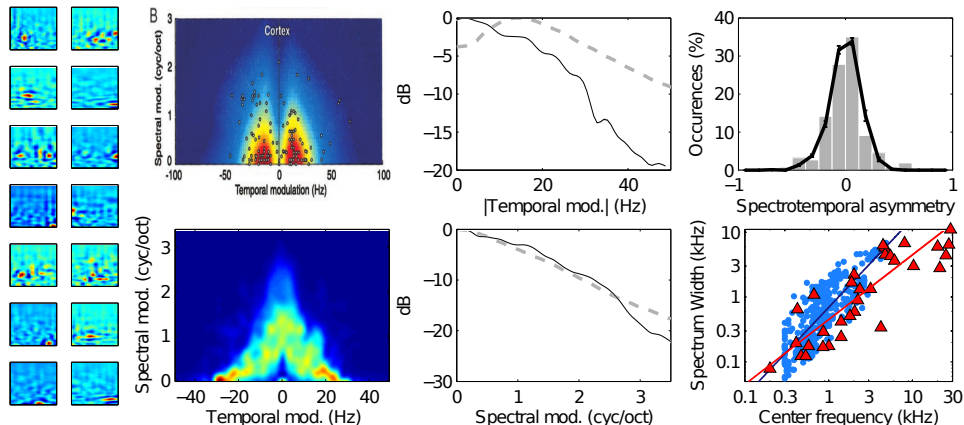

Figure 2: Comparison to A1. Left: RBM bases. Second from left, top: Composite MTF in cat A1, reproduced from [16]. Bottom: Composite MTF for RBM. Second from right, top: temporal MTF in A1 (dashed gray) and for our model (black). Bottom: spectral MTF. Right, top: frequency sweep preference. Bottom: Spectrum width vs center frequency for A1 neurons (red triangles) and model neurons (blue circles).

## 3.2    Primary auditory cortex

In contrast to the large amount of work in the visual system, few efficient coding studies have addressed response properties in primary auditory cortex (but see [20]). We base our comparison on natural sound data consisting of a mixture of data from the Pittsburgh Natural Sounds database and the TIMIT speech corpus. A mix of speech and natural sounds was reported to be necessary to achieve a good match to auditory nerve fiber responses in previous sparse coding work [4]. We transform the raw sound waveform into a representation of its frequency content over time meant to approximate the response of the cochlea [21]. In particular, we pass the input sound signal to a gammatone filterbank which approximates auditory nerve fiber responses [21]. The energy of the filter responses is then summed within fixed time-bins at regular intervals, yielding a representation similar to a spectrogram. We then whiten the data to model subcortical processing. Although there is evidence for temporal whitening in the responses of afferents to auditory cortex, this is certainly a very poor aproximation of subcortical auditory processing [16]. After whitening, we applied unsupervised learning models, yielding the bases shown in Fig. 2 for RBMs. These bases map from our spectrogram input to the model neuron output, and hence represent the spectrotemporal receptive field (STRF) of the model neurons.

We then compared properties of our model STRFs to those measured in cortex. First, based on the experiments reported in O'Connor et al. [22], we analyze the relationship between spectrum bandwidth and center frequency. O'Connor et al. found a nearly linear relationship between these, which matches well with the scaling seen in our model bases (see Fig. 2 bottom right). Next we compared model receptive fields to the composite cortical modulation transfer function reported in [16]. The modulation transfer function (MTF) of a neuron is the amplitude of the 2D Fourier transform of its STRF. The STRF contains one spectral and one temporal axis, and hence its 2D Fourier transform contains one spectral modulation and one temporal modulation axis. The composite MTF is the average of the MTFs computed for each neuron, and for all five algorithms it has a characteristic inverted "V" shape evident in Fig. 2. Summing the composite MTF over time yields the spectral MTF, which is low-pass for our models and well-matched to the spectral MTF reported in cat A1[16]. Summing over the spectral dimension yields the temporal MTF, which is low-pass in our models but band-pass in the experimental data. Finally, we investigate the preference of neurons for upsweeps in frequency versus downsweeps, which can be cast in terms of the MTF by measuring the energy in the left half compared to the right half. The difference in these energies normalized by their sum is the spectrotemporal asymmetry, shown in Fig. 2 top right. All algorithms showed qualitatively similar distributions of spectrotemporal asymmetry to that found in cat A1. Hence the model bases are broadly consistent with receptive field properties measured in primary auditory cortex such as a roughly linear scaling of center frequency with spectrum bandwidth; a low-pass

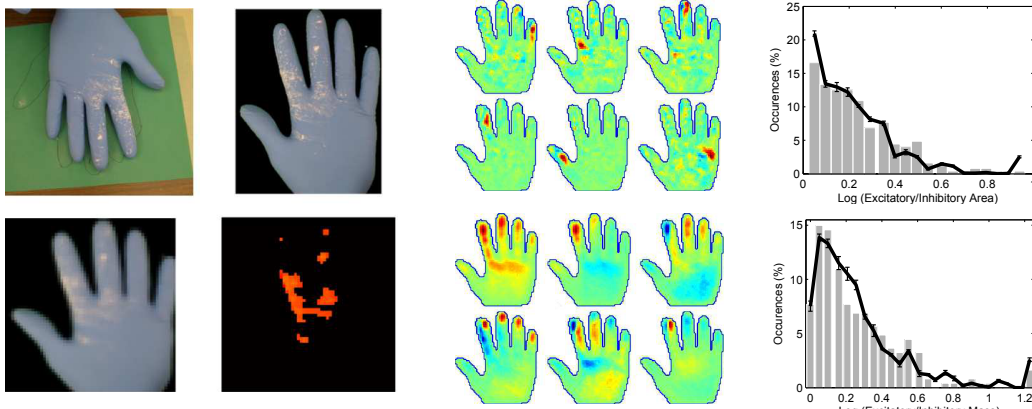

Figure 3: Left: Data collection pipeline. Center: Top two rows, sparse autoencoder bases. Bottom two rows, first six PCA components. Right: Histograms of receptive field structure for the sparse autoencoder algorithm. Black, model distribution. Gray, experimental data from [17]. (Best viewed in color)

spectral MTF of appropriate slope; and a similar distribution of spectrotemporal asymmetry. The models differ from experiment in their temporal structure, which is band-pass in the experimental data but low-pass in our models.

### 3.3 Primary somatosensory cortex

Finally, we test whether these learning algorithms can model somatosensory receptive fields on the hand. To enable this comparison we collected a naturalistic somatosensory dataset meant to capture the statistics of contact points on the hand during normal primate grasping behavior. A variety of objects were dusted with fine white powder and then grasped by volunteers wearing blue latex gloves. To match the natural statistics of primate grasps, we performed the same grip types in the same proportions as observed ecologically in a study of semi-free ranging *Macaca mulatta* [23]. Points of contact were indicated by the transfer of powder to the gloved hand, which was then placed palm-up on a green background and imaged using a digital camera. The images were then post-processed to yield an estimate of the pressure applied to the hand during the grasp (Fig. 3, left).

The dataset has a number of limitations: it contains no temporal information, but rather records all areas of contact for the duration of the grip. Most significantly, it contains only 1248 individual grasps due to the high effort required to collect such data (∼4 minutes/sample), and hence is an order of magnitude smaller than the datasets used for the vision and auditory analyses. Given these limitations, we decided to compare our receptive fields to those found in area 3b of primary somatosensory cortex. Neurons in area 3b respond to light cutaneous stimulation of restricted regions of glabrous skin [24], the same sort of contact that would transfer powder to the glove. Area 3b neurons also receive a large proportion of inputs from slowly adapting mechanoreceptor afferents with sustained responses to static skin indentation [25], making the lack of temporal information less problematic.

Bases learned by the algorithms are shown in Fig. 3. These exhibit a number of qualitative features that accord with the biology. As in area 3b, the model receptive fields are localized to a single digit [24], and receptive field sizes are larger on the palm than on the fingers [25]. These qualitative features are not shared by PCA bases, which typically span multiple fingers. As a more quantitative assesment, we compared model receptive fields on the finger tips to those derived for area 3b neurons in [17]. We computed the ratio between excitatory and inhibitory area for each basis, and plot a population histogram of this ratio, shown for the sparse autoencoder algorithm in the right panel of Fig. 3. Importantly, because this comparison is based on the ratio of the areas, it is not affected by the unknown scale factor between the dimensions of our glove images and those of the macaque hand. We also plot the ratio of the excitatory and inhibitory mass, where excitatory and inhibitory mass is defined as the sum of the positive and negative coefficients in the receptive field, respectively. We find good agreement for all the algorithms we tested.

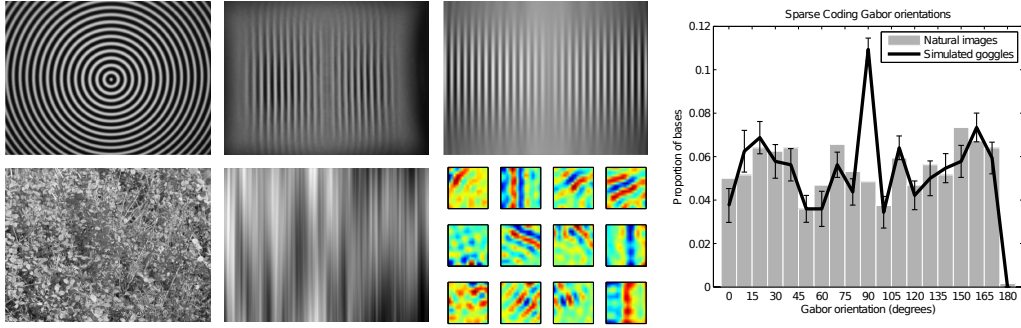

Figure 4: Top row: Input image; Resulting goggle image, reproduced from [26]; Our simulated goggle image. Bottom row: Natural image; Simulated goggle image; Bases learned by sparse coding. Right: Orientation histogram for model neurons is biased towards goggle orientation ($90°$).

# 4 Adaptation to altered environmental statistics

Numerous studies in multiple sensory areas and species document plasticity of receptive field properties in response to various experimental manipulations during an organism's lifetime. In visual cortex, for instance, orientation selectivity can be altered by rearing animals in unidirectionally oriented environments [26]. In auditory cortex, pulsed-tone rearing results in an expansion in the area of auditory cortex tuned to the pulsed tone frequency [27]. And in somatosensory cortex, surgically fusing digits 3 and 4 (the middle and ring fingers) of the hand to induce an artificial digital syndactyly (webbed finger) condition results in receptive fields that span these digits [28]. In this section we ask whether the same learning algorithms that explain features of normal receptive fields can also explain these alterations in receptive field properties due to manipulations of sensory experience.

## 4.1 Goggle-rearing alters V1 orientation tuning

The preferred orientations of neurons in primary visual cortex can be strongly influenced by altering visual inputs during development; Tanaka et al. fitted goggles that severely restricted orientation information to kittens at postnatal week three, and documented a massive overrepresentation of the goggle orientation subsequently in primary visual cortex [26]. Hsu and Dayan [29] have shown that an unsupervised learning algorithm, the product-of-experts model (closely related to ICA), can reproduce aspects of the goggle-rearing experiment. Here we follow their methods, extending the analysis to the other four algorithms we consider.

To simulate the effect of the goggles on an input image, we compute the 2D Fourier transform of the image and remove all energy except at the preferred orientation of the goggles. We slightly blur the resulting image with a small Gaussian filter. Because the kittens receive some period of natural experience, we trained the models on mixtures of patches from natural and altered images, adding one parameter in addition to the algorithmic parameters. Fig. 4 shows resulting receptive fields obtained using the sparse coding algorithm. After learning, the preferred orientations of the bases were derived using the analysis described in Section 3.1. All five algorithms demonstrated an overrepresentation of the goggle orientation, consistent with the experimental data.

## 4.2 Pulsed-tone rearing alters A1 frequency tuning

Early sensory experience can also profoundly alter properties of neural receptive fields in primary auditory cortex. Along similar lines to the results for V1 in Section 4.1, early exposure to a pulsed tone can induce shifts in the preferred center frequency of A1 neurons. In particular, de Villers-Sidani et al. raised rats in an environment with a free field speaker emitting a tone with 40Hz amplitude modulation that repeatedly cycled on for 250ms then off for 500ms [27]. Mapping the preferred center frequencies of neurons in tone-exposed rats revealed a corresponding overrepresentation in A1 around the pulsed-tone frequency.

We instantiated this experimental paradigm by adding a pulsed tone to the raw sound waveforms of the natural sounds and speech before computing the gammatone responses. Example bases for ICA are shown in the center panel of Fig. 5, many of which are tuned to the pulsed-tone frequency. We computed the preferred frequency of each model receptive field by summing the square of each patch along the temporal dimension. The right panel of Fig. 5 shows population histograms of the

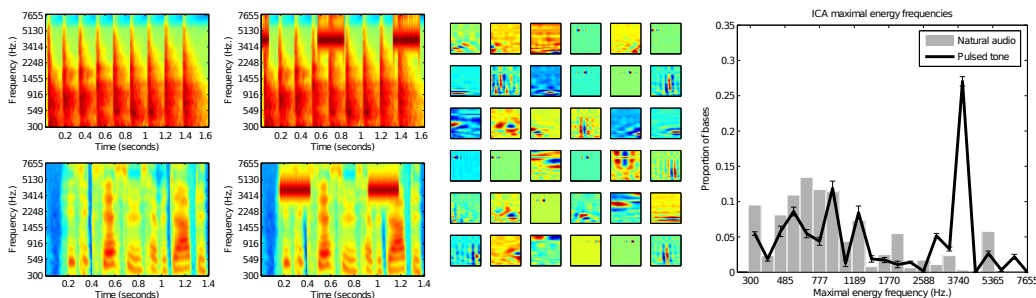

Figure 5: Left: Example spectrograms before and after adding a 4kHz pulsed tone. Center: ICA bases learned from pulsed tone data. Right: Population histograms of preferred frequency reveal a strong preference for the pulsed-tone frequency of 4kHz.

preferred center frequencies for models trained on natural and pulsed-tone data for ICA and K-means. We find that all algorithms show an overrepresentation in the frequency band containing the tone, in qualitative agreement with the results reported in [27]. Intuitively, this overrepresentation is due to the fact that many bases are necessary to represent the temporal information present in the pulsed-tone, that is, the phase of the amplitude modulation and the onset or offset time of the stimulus.

### 4.3 Artificial digital syndactyly in S1

Allard et al. [28] surgically fused adjacent skin on digits 3 and 4 in adult owl monkeys to create an artificial sydactyly, or webbed finger, condition. After 14, 25, or 33 weeks, many receptive fields of neurons in area 3b of S1 were found to span digits 3 and 4, a qualitative change from the normally strict localization of receptive fields to a single digit. Additionally, at the tips of digits 3 and 4 where there is no immediately adjacent skin on the other digit, some neurons showed discontinuous double-digit receptive fields that responded to stimulation on either finger tip [28]. In contrast to the shifts in receptive field properties described in the preceding two sections, these striking changes are qualitatively different, and as such provide an important test for functional models of plasticity.

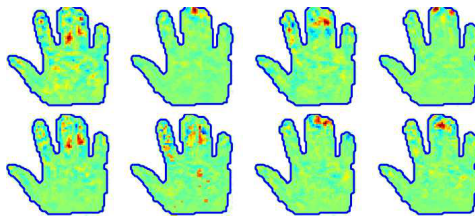

Figure 6: Bases trained on artificial syndactyly data. Top row: Sparse coding. Bottom row: K-means.

We modeled the syndactyly condition by fusing digits 3 and 4 of our gloves and collecting 782 additional grip samples according to the method in Section 3.3. Bases learned from this syndactyly dataset are shown in Fig. 4.3. All models learned double-digit receptive fields that spanned digits 3 and 4, in qualitative agreement with the findings reported in [28]. Additionally, a small number of bases contained discontinuous double-digit receptive fields consisting of two well-separated excitatory regions on the extreme finger tips (e.g., Fig. 4.3 top right). In contrast to the experimental findings, model receptive fields spanning digits 3 and 4 also typically have a discontinuity along the seam. We believe this reflects a limitation of our dataset; although digits 3 and 4 of our data collection glove are fused together and must move in concert, the seam between these digits remains inset from the neighboring fingers, and hence grasps rarely transfer powder to this area. In the experiment, the skin was sutured to make the seam flush with the neighboring fingers.

## 5 Discussion

Taken together, our results demonstrate that a number of unsupervised learning algorithms can account for certain normal and altered linear receptive field properties across multiple primary sensory cortices. Each of the five algorithms we tested obtained broadly consistent fits to experimental data in V1, A1 and S1. Although these fits were not perfect–notably, missing "blob" receptive fields in V1 and bandpass temporal structure in A1–they demonstrate the feasibility of applying a single learning algorithm to experimental data from multiple modalities.

In no setting did one of our five algorithms yield qualitatively different results from any other. This finding likely reflects the underlying similarities between the algorithms, which all attempt to find a sparse representation of the input while preserving information about it. The relative robustness of our results to the details of the algorithms offers one explanation of the empirical observation of similar plasticity outcomes at a functional level despite potentially very different underlying mechanisms [8]. Even if the mechanisms differ, provided that they still incorporate some version of sparsity, they can produce qualitatively very similar outcomes.

The success of these models in capturing the effects of experimental manipulations of sensory input suggests that the adaptation of receptive field properties to natural statistics, as proposed by efficient coding models, may occur significantly on developmental timescales. If so, this would allow the extensive literature on plasticity to constrain further modeling efforts.

Furthermore, the ability of a single algorithm to capture responses in multiple sensory cortices shows that, in principle, a qualitatively similar plasticity process could operate throughout primary sensory cortices. Experimentally, such a possibility has been addressed most directly by cortical "rewiring" experiments, where visual input is rerouted to either auditory or somatosensory cortex [30, 31, 32, 33, 34, 35]. In neonatal ferrets, visual input normally destined for lateral geniculate nucleus can be redirected to the auditory thalamus, which then projects to primary auditory cortex. Roe et al. [32] and Sharma et al. [34] found that rewired ferrets reared to adulthood had neurons in auditory cortex responsive to oriented edges, with orientation tuning indistinguishable from that in normal V1. Further, Von Melchner et al. [33] found that rewired auditory cortex can mediate behavior such as discriminating between different grating stimuli and navigating toward a light source. Rewiring experiments in hamster corroborate these results, and in addition show that rewiring visual input to somatosensory cortex causes S1 to exhibit light-evoked responses similar to normal V1 [31, 35]. Differences between rewired and normal cortices do exist–for example, the period of the orientation map is larger in rewired animals [34]. However, these experiments are consistent with the hypothesis that sensory cortices share a common learning algorithm, and that it is through activity dependent development that they specialize to a specific modality. Our results provide a possible explanation of these experiments, as we have shown constructively that the exact same algorithm can produce V1-, A1-, or S1-like receptive fields depending on the type of input data it receives.

**Acknowledgements** We give warm thanks to Andrew Maas, Cynthia Henderson, Daniel Hawthorne and Conal Sathi for code and ideas. This work is supported by the DARPA Deep Learning program under contract number FA8650-10- C-7020. Andrew Saxe is supported by a NDSEG and Stanford Graduate Fellowship.

## Footnotes

[1]Taking the log of the image intensities before whitening, as in [3], yielded similar fits to V1 data.

# References

[1] V.B. Mountcastle. *An organizing principle for cerebral function: The unit module and the distributed system.*, pages 7–50. MIT Press, Cambridge, MA, 1978.

[2] B.A. Olshausen and D.J. Field. Emergence of simple-cell receptive field properties by learning a sparse code for natural images. *Nature*, 381(6583):607–9, 1996.

[3] J.H. van Hateren and D.L. Ruderman. Independent component analysis of natural image sequences yields spatio-temporal filters similar to simple cells in primary visual cortex. *Proc. R. Soc. Lond. B*, 265(1412):2315–20, December 1998.

[4] E.C. Smith and M.S. Lewicki. Efficient auditory coding. *Nature*, 439(7079):978–82, 2006.

[5] D.L. Ringach. Spatial structure and symmetry of simple-cell receptive fields in macaque primary visual cortex. *J. Neurophysiol.*, 88(1):455–63, July 2002.

[6] M. Rehn and F.T. Sommer. A network that uses few active neurones to code visual input predicts the diverse shapes of cortical receptive fields. *J. Comput. Neurosci.*, 22(2):135–46, April 2007.

[7] G. Puertas, J. Bornschein, and J. Lucke. The Maximal Causes of Natural Scenes are Edge Filters. In *NIPS*, 2010.

[8] D.E. Feldman. Synaptic mechanisms for plasticity in neocortex. *Annu. Rev. Neurosci.*, 32:33–55, January 2009.

[9] K. Fox and R.O.L. Wong. A comparison of experience-dependent plasticity in the visual and somatosensory systems. *Neuron*, 48(3):465–77, November 2005.

[10] A.J. Bell and T.J. Sejnowski. The "independent components" of natural scenes are edge filters. *Vision Res.*, 37(23):3327–38, December 1997.

[11] P. Vincent, H. Larochelle, Y. Bengio, and P. Manzagol. Extracting and Composing Robust Features with Denoising Autoencoders. In *ICML*, 2008.

[12] H. Lee, C. Ekanadham, and A.Y. Ng. Sparse deep belief net model for visual area V2. In *NIPS*, 2008.

[13] A. Coates, H. Lee, and A.Y. Ng. An Analysis of Single-Layer Networks in Unsupervised Feature Learning. In *AISTATS*, 2011.

[14] R.L. De Valois, D.G. Albrecht, and L.G. Thorell. Spatial frequency selectivity of cells in macaque visual cortex. *Vision Res.*, 22(5):545–59, January 1982.

[15] R.L. De Valois, E.W. Yund, and N. Hepler. The orientation and direction selectivity of cells in macaque visual cortex. *Vision Res.*, 22(5):531–544, 1982.

[16] L.M. Miller, M.A. Escabí, H.L. Read, and C.E. Schreiner. Spectrotemporal receptive fields in the lemniscal auditory thalamus and cortex. *J. Neurophysiol.*, 87(1):516–27, January 2002.

[17] J.J. DiCarlo, K.O. Johnson, and S.S. Hsiao. Structure of receptive fields in area 3b of primary somatosensory cortex in the alert monkey. *J. Neurosci.*, 18(7):2626–45, April 1998.

[18] M. Carandini, J.B. Demb, V. Mante, D.J. Tolhurst, Y. Dan, B.A. Olshausen, J.L. Gallant, and N.C. Rust. Do we know what the early visual system does? *J. Neurosci.*, 25(46):10577–97, November 2005.

[19] A. Hyvärinen, J. Hurri, and P.O. Hoyer. *Natural Image Statistics*. Springer, London, 2009.

[20] D.J. Klein, P. König, and K.P. Körding. Sparse Spectrotemporal Coding of Sounds. *EURASIP J. Adv. Sig. Proc.*, 7:659–667, 2003.

[21] R.D. Patterson, K. Robinson, J. Holdsworth, D. McKeown, C. Zhang, and M. Allerhand. Complex sounds and auditory images. In *Adv. Biosci.*, 1992.

[22] K.N. O'Connor, C.I. Petkov, and M.L. Sutter. Adaptive stimulus optimization for auditory cortical neurons. *J. Neurophysiol.*, 94(6):4051–67, 2005.

[23] N.B.W. Macfarlane and M.S.A. Graziano. Diversity of grip in Macaca mulatta. *Exp. Brain Res.*, 197(3):255–68, August 2009.

[24] M. Sur. Receptive fields of neurons in areas 3b and 1 of somatosensory cortex in monkeys. *Brain Res.*, 198(2):465–471, October 1980.

[25] R.L. Paul, M.M. Merzenich, and H. Goodman. Representation of slowly and rapidly adapting cutaneous mechanoreceptors of the hand in Brodmann's areas 3 and 1 of Macaca mulatta. *Brain Res.*, 36(2):229–49, January 1972.

[26] S. Tanaka, J. Ribot, K. Imamura, and T. Tani. Orientation-restricted continuous visual exposure induces marked reorganization of orientation maps in early life. *NeuroImage*, 30(2):462–77, April 2006.

[27] E. de Villers-Sidani, E.F. Chang, S. Bao, and M.M. Merzenich. Critical period window for spectral tuning defined in the primary auditory cortex (A1) in the rat. *J. Neurosci.*, 27(1):180–9, 2007.

[28] T. Allard, S.A. Clark, W.M. Jenkins, and M.M. Merzenich. Reorganization of somatosensory area 3b representations in adult owl monkeys after digital syndactyly. *J. Neurophysiol.*, 66(3):1048–58, September 1991.

[29] A.S. Hsu and P. Dayan. An unsupervised learning model of neural plasticity: Orientation selectivity in goggle-reared kittens. *Vision Res.*, 47(22):2868–77, October 2007.

[30] M. Sur, P. Garraghty, and A. Roe. Experimentally induced visual projections into auditory thalamus and cortex. *Science*, 242(4884):1437–1441, December 1988.

[31] C. Métin and D.O. Frost. Visual responses of neurons in somatosensory cortex of hamsters with experimentally induced retinal projections to somatosensory thalamus. *PNAS*, 86(1):357–61, January 1989.

[32] A.W. Roe, S.L. Pallas, Y.H. Kwon, and M. Sur. Visual projections routed to the auditory pathway in ferrets: receptive fields of visual neurons in primary auditory cortex. *J. Neurosci.*, 12(9):3651–64, September 1992.

[33] L. von Melchner, S.L. Pallas, and M. Sur. Visual behaviour mediated by retinal projections directed to the auditory pathway. *Nature*, 404(6780):871–876, 2000.

[34] J. Sharma, A. Angelucci, and M. Sur. Induction of visual orientation modules in auditory cortex. *Nature*, 404(April):841–847, 2000.

[35] D.O. Frost, D. Boire, G. Gingras, and M. Ptito. Surgically created neural pathways mediate visual pattern discrimination. *PNAS*, 97(20):11068–73, September 2000.

